# The topographic unsupervised learning of natural sounds in the auditory cortex

**Hiroki Terashima**
The University of Tokyo / JSPS
Tokyo, Japan
teratti@teratti.jp

**Masato Okada**
The University of Tokyo / RIKEN BSI
Tokyo, Japan
okada@k.u-tokyo.ac.jp

## Abstract

The computational modelling of the primary auditory cortex (A1) has been less fruitful than that of the primary visual cortex (V1) due to the less organized properties of A1. Greater disorder has recently been demonstrated for the tonotopy of A1 that has traditionally been considered to be as ordered as the retinotopy of V1. This disorder appears to be incongruous, given the uniformity of the neocortex; however, we hypothesized that both A1 and V1 would adopt an efficient coding strategy and that the disorder in A1 reflects natural sound statistics. To provide a computational model of the tonotopic disorder in A1, we used a model that was originally proposed for the smooth V1 map. In contrast to natural images, natural sounds exhibit distant correlations, which were learned and reflected in the disordered map. The auditory model predicted harmonic relationships among neighbouring A1 cells; furthermore, the same mechanism used to model V1 complex cells reproduced nonlinear responses similar to the pitch selectivity. These results contribute to the understanding of the sensory cortices of different modalities in a novel and integrated manner.

## 1   Introduction

Despite the anatomical and functional similarities between the primary auditory cortex (A1) and the primary visual cortex (V1), the computational modelling of A1 has proven to be less fruitful than V1, primarily because the responses of A1 cells are more disorganized. For instance, the receptive fields of V1 cells are localized within a small portion of the field of view [1], whereas certain A1 cells have receptive fields that are not localized, as these A1 cells demonstrate significant responses to multiple distant frequencies [2, 3]. An additional discrepancy that has recently been discovered between these two regions relates to their topographic structures, i.e., the retinotopy of V1 and the tonotopy of A1; these structures had long been considered to be quite similar, but studies on a microscopic scale have demonstrated that in mice, the tonotopy of A1 is much more disordered [4, 5] than the retinotopy of V1 [6, 7]. This result is consistent with previous investigations involving other species [8, 9], suggesting that the discrepancy in question constitutes a general tendency among mammals. This disorderliness appears to pose significant difficulties for the development of computational models of A1.

A number of computational modelling studies have emphasized the close associations between V1 cells and natural image statistics, which suggests that the V1 adopts an unsupervised, efficient coding strategy [10]. For instance, the receptive fields of V1 simple cells were reproduced by either sparse coding [11] or the independent component analysis [12] of natural images. This line of research yields explanations for the two-dimensional topography, the orientation and retinotopic maps of V1 [13, 14, 15]. Similar efforts to address A1 have been attempted by only a few studies, which demonstrated that the efficient coding of natural, harmonic sounds, such as human voices or piano

recordings, can explain the basic receptive fields of A1 cells [16, 17] and their harmony-related responses [18, 19]. However, these studies have not yet addressed the topography of A1.

In an integrated and computational manner, the present paper attempts to explain why the tonotopy of A1 is more disordered than the retinotopy of V1. We hypothesized that V1 and A1 still share an efficient coding strategy, and we therefore proposed that the distant correlations in natural sounds would be responsible for the relative disorder in A1. To test this hypothesis, we first demonstrated the significant differences between natural images and natural sounds. Natural images and natural sounds were then each used as inputs for topographic independent component analysis, a model that had previously been proposed for the smooth topography of V1, and maps were generated for these images and sounds. Due to the distant correlations of natural sounds, greater disorder was observed in the learned map that had been adapted to natural sounds than in the analogous map that had been adapted to images. For natural sounds, this model not only predicted harmonic relationships between neighbouring cells but also demonstrated nonlinear responses that appeared similar to the responses of the pitch-selective cells that were recently found in A1. These results suggest that the apparently dissimilar topographies of V1 and A1 may reflect statistical differences between natural images and natural sounds; however, these two regions may employ a common adaptive strategy.

## 2    Methods

### 2.1    Topographic independent component analysis

Herein, we discuss an unsupervised learning model termed topographic independent component analysis (TICA), which was originally proposed for the study of V1 topography [13, 14]. This model comprises two layers: the first layer of $N$ units models the linear responses of V1 simple cells, whereas the second layer of $N$ units models the nonlinear responses of V1 complex cells, and the connections between the layers define a topography. Given a whitened input vector $\boldsymbol{I}(x) \in \mathbb{R}^d$ (here, $d = N$), the input is reconstructed by the linear superposition of a basis $\boldsymbol{a}_i \in \mathbb{R}^d$, each of which corresponds to the first-layer units

$$\boldsymbol{I} = \sum_i s_i \boldsymbol{a}_i \tag{1}$$

where $s_i \in \mathbb{R}$ are activity levels of the units or model neurons. Inverse filters $\boldsymbol{w}_i$ to determine $s_i$ can typically be obtained, and thus $s_i = \boldsymbol{I}^T \boldsymbol{w}_i$ (inner product). Using the activities of the first layer, the activities of the second-layer units $c_i \in \mathbb{R}$ can be defined as follows:

$$c_i = \sum_j h(i, j) s_j^2 \tag{2}$$

where $h(i, j)$ is the neighbourhood function that takes the value of 1 if $i$ and $j$ are neighbours and is 0 otherwise. The neighbourhood is defined by a square window (e.g., $5 \times 5$) in cases of two-dimensional topography. The learning of $\boldsymbol{w}_i$ is accomplished through the minimization of the energy function $E$ or the negative log likelihood:

$$E = -\log L(\boldsymbol{I}; \{\boldsymbol{w}_i\}) = -\sum_i G(c_i) \tag{3}$$

$$\Delta \boldsymbol{w}_i \propto \left\langle \boldsymbol{I} s_i \left( \sum_j h(i, j) g(c_j) \right) \right\rangle \tag{4}$$

where $G(c_i) = -\sqrt{\epsilon + c_i}$ imposes sparseness on the second-layer activities ($\epsilon = 0.005$ for the stability), and $g(c_i)$ is the derivative of $G(c_i)$. The operator $\langle \cdots \rangle$ is the mean over the iterations.

#### 2.1.1    An extension for overcomplete representation

Ma and Zhang [15] extended the TICA model to account for overcomplete representations ($d < N$), which are observed in the V1 of primates. In this extension, inverse filters cannot be uniquely defined; therefore, a set of first-layer responses $s_i$ to an input is computed by minimizing the following extended energy function:

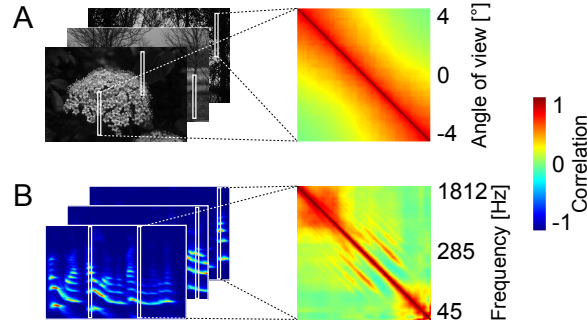

Figure 1: **Local correlations in natural images and distant correlations in natural sounds.** (*A*) The correlation matrix of image strips (right) demonstrated only local correlations ($\sim 6°$) in the field of view ($\sim 120°$). (*B*) The correlation matrix of the human voice spectra (right) demonstrated not only local correlation but also off-diagonal distant correlations produced by harmonics.

$$E = -\log L(\boldsymbol{I}; \{\boldsymbol{a}_i\}, \{s_i\}) = \left| \boldsymbol{I} - \sum_i s_i \boldsymbol{a}_i \right|^2 - \lambda \sum_i G(c_i) \tag{5}$$

$$\Delta s_i \propto \boldsymbol{a}_i^T \left( \boldsymbol{I} - \sum_j s_j \boldsymbol{a}_j \right) - \lambda s_i \left( \sum_j h(i,j) g(c_j) \right) \tag{6}$$

where $\lambda$ is the relative weight of the activity sparseness, in accordance with sparse coding [11]. The initial value of $s_i$ is set equal to the inner product of $\boldsymbol{I}$ and $\boldsymbol{a}_i$. Every 256 inputs, the basis is updated using the following gradient. In this study, we used the learning rate $\eta = 0.08$.

$$\Delta \boldsymbol{a}_i = \eta \left\langle s_i \left( \boldsymbol{I} - \sum_j s_j \boldsymbol{a}_j \right) \right\rangle \tag{7}$$

### 2.2 The discontinuity index for topographic representation

To compare the degrees of disorder in topographies of different modalities, we defined a discontinuity index (DI) for each point $i$ of the maps. Features defining a topography $f(i)$ (e.g., a retinotopic position or a frequency) were normalized to the range of $[0, 1]$. Features $f(j)$ within the neighbourhood of the $i$th unit defined by $h(i, j)$ were linearly fitted using the least squares method, and the DI value at $i$ was then determined using the following equation:

$$\mathrm{DI}(i) = \sqrt{\frac{\sum_j h(i,j) r(j)^2}{N_{\mathrm{NB}}}} \tag{8}$$

where $r(j)$ is the residual error of linear regression at $j$ and $N_{\mathrm{NB}}$ is the number of units within a neighbourhood window. If the input space is a torus (see Section 3.3), another DI value is computed using modified $f$ values that are increased by 1 if they were initially within $[0, \frac{1}{2})$, and the smaller of the calculated DI values is used.

## 3 Results

### 3.1 Correlations of natural images and natural sounds

Given that V1 is supposed to adapt to natural images and that A1 is supposed to adapt to natural sounds, the first analysis in this study simply compared statistics for natural images and natural sounds. The natural images were taken from the van Hateren database [20] and were reduced four times from their original size. Vertical arrays of 120 pixels each were extracted from the reduced

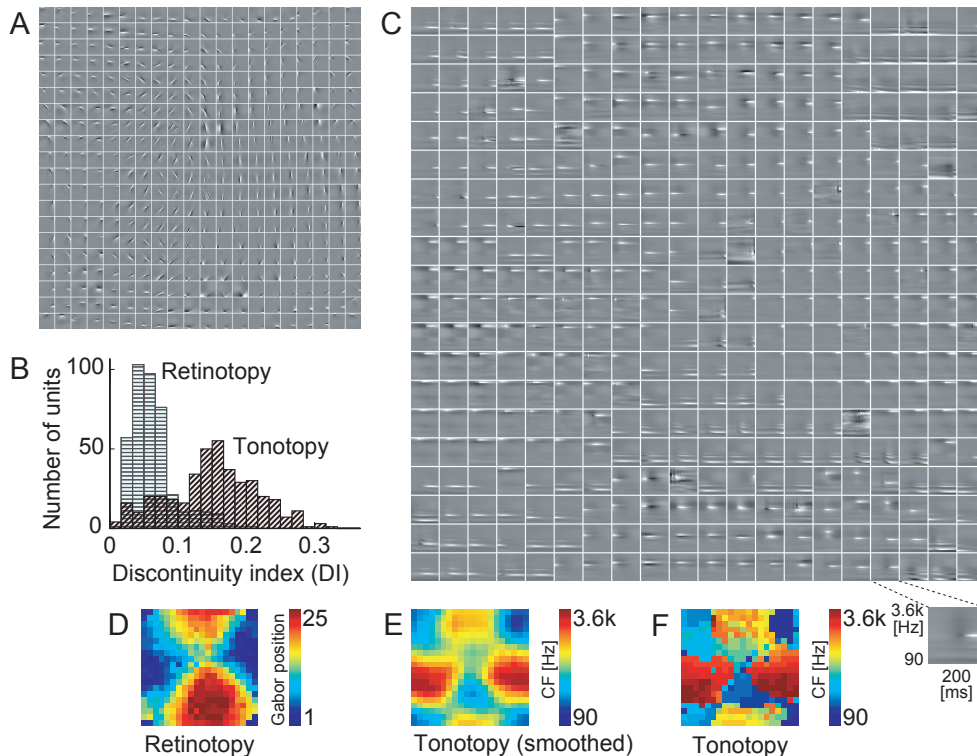

Figure 2: **The ordered retinotopy and disordered tonotopy.** (*A*) The topography of units adapted to natural images. A small square indicates a unit $a_i$ (grey: 0; white: max value). (*B*) The distributions of DI for the two topographies. (*C*) The topography of spectro-temporal units that have been adapted to natural sounds. (*D-F*) The retinotopy of the visual map (D) is smooth, whereas the tonotopy of the auditory map (F) is more disordered, although global tonotopy still exists (E).

images, each of which covered approximately $8°$ ($\frac{1}{15}$ of the vertical range of the human field of view). Figure 1A (right) illustrates the correlation matrix for these images, which is a simple structure that contains local correlations that span approximately $6°$. This result was not surprising, as distant pixels typically depict different objects.

For natural sounds, we used human narratives from the Handbook of the International Phonetic Association [21], as efficient representations of human voices have been successful in facilitating studies of various components of the auditory system [22, 23], including A1 [16, 17]. After these sounds were downsampled to 4 kHz, their spectrograms were generated using the NSL toolbox [24] to approximate peripheral auditory processing. Short-time spectra were extracted from the spectrograms, each of which were 128 pixels wide on a logarithmic scale (24 pixels = 1 octave). Note that the frequency range ($> 5$ octaves) spans approximately half of a typical mammalian hearing range ($\sim 10$ octaves [25]), whereas the image pixel array spans only $\frac{1}{15}$ of the field of view.

Figure 1B illustrates the correlation matrix for these sounds, which is a complex structure that incorporates distant, off-diagonal correlations. The most prominent off-diagonal correlation, which was just 1 octave away from the main diagonal, corresponded to the second harmonic of a sound, i.e., frequencies at a ratio of 1:2. Similarly, other off-diagonal peaks indicated correlations due to higher harmonics, i.e., frequencies that were related to each other by simple integral ratios. These distant correlations represent relatively typical results for natural sounds and differ greatly from the strictly local correlations observed for natural images.

## 3.2 Greater disorder for the tonotopy than the retinotopy

To test the hypothesis that V1 and A1 share a learning strategy, the TICA model was applied to natural images and natural sounds, which exhibit different statistical profiles, as discussed above.

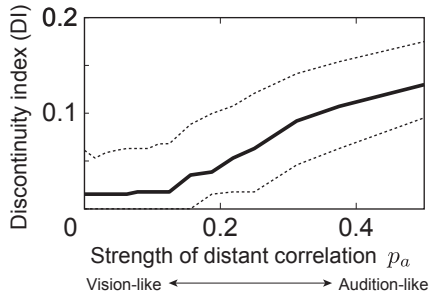

Figure 3: **The correlation between discontinuity and input "auditoriness".** When inputs only correlated locally ($p_a \sim 0$: vision-like inputs), DI was low, and DI increased with the input "auditoriness" $p_a$. Three lines: the quartiles (25, 50 (bold), and 75%) obtained from 100 iterations.

Learning with natural images was accomplished in accordance with the original TICA study [13, 14]. Images from the van Hateren database were reduced four times from their original size, and $25 \times 25$ pixel image patches were randomly extracted (n = 50,000). The patches were whitened and bandpassed by applying principal component analysis, whereby we selected 400 components and rejected certain components with low variances and the three components with the largest variances [13]. The topography was a $20 \times 20$ torus, and the neighbourhood window was $5 \times 5$.

Figure 2A illustrates the visual topographic map obtained from this analysis, a small square of which constitutes a basis vector $a_i$. As previously observed in the original TICA study [13, 14], each unit was localized, oriented, and bandpassed; thus, these units appeared to be organized similarly to the receptive fields of V1 simple cells. The orientation and position of the units changed smoothly with the coordinates that were examined, which suggested that this map evinces an ordered topography. To quantify the retinotopic discontinuity, each unit was fitted using a two-dimensional Gabor function, and DI was calculated using the y values of the centre coordinates of the resulting Gabor functions as the features. Figure 2B graphically indicates that the obtained DI values were quite low, which is consistent with the smooth retinotopy illustrated in Figure 2D.

Next, another TICA model was applied to natural sounds to create an auditory topographic map that could be compared to the visual topography. As detailed in the previous section, spectrograms of human voices (sampled at 8 kHz) were generated using the NSL toolbox to approximate peripheral auditory processing. Spectrogram patches of 200 ms (25 pixels) in width were randomly extracted (n = 50,000) and vertically reduced from 128 to 25 pixels, which enabled these spectrogram patches to be directly compared with the image patches. The sound patches were whitened, bandpassed, and adapted using the model in the same manner as was described for the image patches.

Figure 2C shows the resulting auditory topographic map, which is composed of spectro-temporal units of $a_i$ that are represented by small squares. The units were localized temporally and spectrally, and some units demonstrated multiple, harmonic peaks; thus, these units appeared to reasonably represent the typical spectro-temporal receptive fields of A1 cells [16, 3]. The frequency to which an auditory neuron responds most significantly is called its characteristic frequency (CF) [2]. In this analysis, the CF of a unit was defined as the frequency that demonstrated the largest absolute value for the unit in question. Figure 2F illustrates the spatial distribution of CFs, i.e., the tonotopic map. Within local regions, the tonotopy was not necessarily smooth, i.e., neighbouring units displayed distant CFs. However, at a global level, a smooth tonotopy was observed (Figure 2E). Both of these findings are consistent with established experimental results [4, 5]. The distribution of tonotopic DI values is shown in Figure 2B, which clearly demonstrates that the tonotopy was more disordered than the retinotopy ($p < 0.0001$; Wilcoxon rank test).

### 3.3 The topographic disorder due to distant input correlations

The previous section demonstrated that natural sounds could induce greater topographic disorder than natural images, and this section discusses the attempts to elucidate the disorder resulting from a specific characteristic of natural sounds, namely, distant correlations. For this purpose, we generated artificial inputs ($d = 16$) with a parameter $p_a \in [0, 1]$ that regulates the degree of distant correlations.

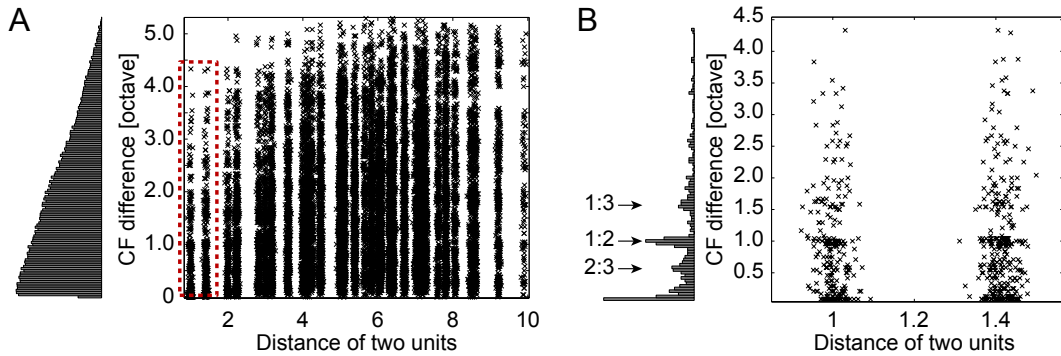

Figure 4: **The harmonic relationships between CFs of neighbouring units.** (*A*) The full distribution of distance and CF difference between two units. (*B*) The distribution of CF differences within neighbourhoods (the red-dotted rectangle in (A)). There were three peaks that indicate harmonic relationships between neighbouring units. The distances were jittered to obtain the visualization.

After the inputs were initially generated from a standard normal distribution, a constant value of $4$ was added at $k$ points of each input, where $k$ was from a uniform distribution over $\{3, 4, 5, 6\}$ and the points' coordinates $x$ were from a normal distribution with a random centre and $\sigma = 2$. After adding this constant value at $x$, we also added another at $x_{dist} = x + 5$ with a probability $p_a$ that defines its "auditoriness", i.e., its degree of distant correlations. For greater simplicity and to avoid border effects, the input space was defined to be a one-dimensional torus. The topography was also set as a one-dimensional torus of 16 units with a neighbourhood window size of 5.

Figure 3 shows the positive correlation between the input "auditoriness" $p_a$ and the DI of the learned topographies. In computations of DI, the feature $f$ of a unit was considered to be its peak coordinate with the largest absolute value, and a toric input space was used (Section 2.2). If the input only demonstrated local correlations like visual stimuli ($p_a \sim 0$), then its learned topography was smooth (i.e., its DI was low). The DI values generally increased as distant correlations appeared more frequently, i.e., more "auditoriness" of the inputs grew. Thus, the topographic disorder of auditory maps results from distant correlations presented by natural auditory signals.

### 3.4 The harmonic relationship among neighbouring units

Several experiments [4, 5] have reported that the CFs of neighbouring cells can differ by up to 4 octaves, although these studies have failed to provide additional detail regarding the local spatial patterns of the CF distributions. However, if the auditory topography is representative of natural stimulus statistics, the topographic map is likely to possess certain additional spatial features that reflect the statistical characteristics of natural sounds.

To enable a detailed investigation of the CF distribution, we employed a model that had been adapted to finer frequency spectra of natural sounds, and this model was then used throughout the remainder of the study. As the temporal structure of the auditory receptive fields was less dominant than their spectral structure (Figure 2C), we focused solely on the spectral domain and did not attempt to address temporal information. Therefore, the inputs for the new model (n = 100,000) were short-time frequency spectra of 128 pixels each (24 pixels = 1 octave). The data for these spectra were first obtained from the spectrograms of human voices (8 kHz) using the method detailed in Section 3.1, and these data were then whitened, bandpassed, and reduced to 100 dimensions prior to input into the model. To illustrate patterns more clearly, the results shown below were obtained using the overcomplete extension of TICA described in Section 2.1.1, which included a $14 \times 14$ torus (approximately $2\times$ overcomplete) and $3\times3$ windows. The CF of a unit was determined using pure-tone inputs of 128 frequencies.

Figure 4A illustrates the full distribution of the distance and CF difference between two units in a learned topography. The CFs of even neighbouring units differed by up to $\sim 4$ octaves, which is consistent with recent experimental findings [4, 5]. A closer inspection of the red-dotted rectangular region of Figure 4A is shown in Figure 4B. The histogram in Figure 4B demonstrates several peaks

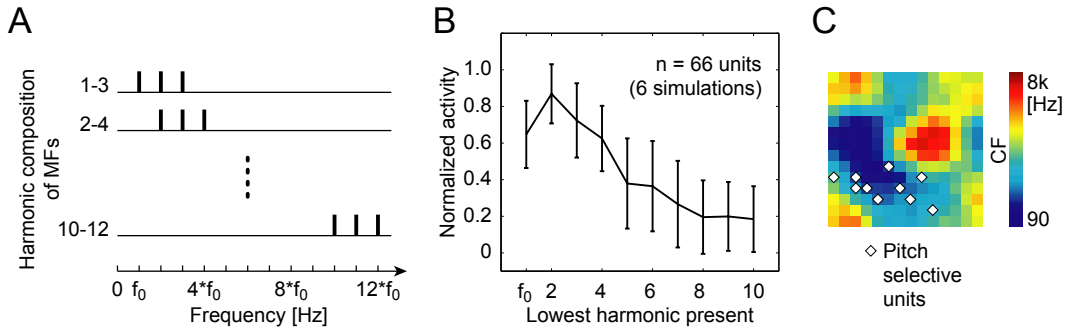

Figure 5: **Nonlinear responses similar to pitch selectivity.** (*A*) The spectra of MFs that share a $f_0$, all of which are perceived similarly. (*B*) The responses of pitch-selective units to MFs. (*C*) The distribution of pitch-selective units on the smoothed tonotopy in a single session.

### 3.5 Nonlinear responses similar to pitch-selectivity

Psychoacoustics have long demonstrated interesting phenomena related to harmony, namely, the perception of pitch, which represents a subjective attribute of perceived sounds. Forming a rigid definition for the notion of pitch is difficult; however, if a tone consists of a stack of harmonics $(f_0, 2f_0, 3f_0, \ldots)$, then its pitch is the frequency of the lowest harmonic, which is called the fundamental frequency $f_0$. The perception of pitch is known to remain constant even if the sound lacks power at lower harmonics; in fact, pitch at $f_0$ can be perceived from a sound that lacks $f_0$, a phenomenon known as "missing fundamental" [26]. Nonlinear pitch-selective responses similar to this perception have recently been demonstrated in certain A1 neurons [27] that localize in the low-frequency area of the global tonotopy.

To investigate pitch-related responses, previously described complex tones [27] that consisted of harmonics were selected as inputs for the model described in Section 3.4. For each unit, responses were calculated to complex tones termed missing fundamental complex tones (MFs) [27]. The MFs were composed of three consecutive harmonics sharing a single $f_0$; the lowest frequency for these consecutive harmonics varied from the fundamental frequency ($f_0$) to the tenth harmonic ($10f_0$), as shown in Figure 5A. For each unit, five patterns of $f_0$ around its CF ($\sim 0.2$ octave) were tested, resulting in a total of $10 \times 5 = 50$ variations of MFs. The activity of a unit was normalized to its maximum response to the MFs. Pitch-selective units were defined as those that significantly responded (normalized activity $> 0.4$) to all of the MFs sharing a single $f_0$ with a lowest harmonic from 1 to 4.

We found certain pitch-selective units in the second layer (n = 66; 6 simulations), whereas none were found in the first layer. Figure 5B illustrates the response profiles of the pitch-selective units, which demonstrated sustained activity for MFs with a lowest harmonic below the sixth harmonic ($6f_0$), and this result is similar to previously published data [27]. Additionally, these units were located in a low-frequency region of the global tonotopy, as shown in Figure 5C, and this feature of pitch-selective units is also consistent with previous findings [27]. The second layer of the TICA model, which contained the pitch-selective units, was originally designed to represent the layer of V1 complex cells, which have nonlinear responses that can be modelled by a summation of "energies" of neighbouring simple cells [13, 14, 15]. Our result suggests that the mechanism underlying V1 complex cells may be similar to the organizational mechanism for A1 pitch-selective cells.

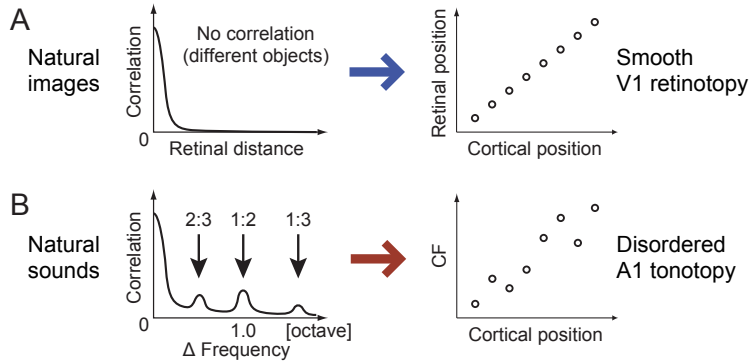

Figure 6: **The suggested relationships between natural stimulus statistics and topographies.**

## 4    Discussion

Using a single model, we have provided a computational account explaining why the tonotopy of A1 is more disordered than the retinotopy of V1. First, we demonstrated that there are significant differences between natural images and natural sounds; in particular, the latter evince distant correlations, whereas the former do not. The topographic independent component analysis therefore generated a disordered tonotopy for these sounds, whereas the retinotopy adapted to natural images was locally organized throughout. Detailed analyses of the TICA model predicted harmonic relationships among neighbouring neurons; furthermore, these analyses successfully replicated pitch selectivity, a nonlinear response of actual cells, using a mechanism that was designed to model V1 complex cells. The results suggest that A1 and V1 may share an adaptive strategy, and the dissimilar topographies of visual and auditory maps may therefore reflect significant differences in the natural stimuli.

Figure 6 summarizes the ways in which the organizations of V1 and A1 reflect these input differences. Natural images correlate only locally, which produces a smooth retinotopy through an efficient coding strategy (Figure 6A). By contrast, natural sounds exhibit additional distant correlations (primarily correlations among harmonics), which produce the topographic disorganization observed for A1 (Figure 6B). To extract the features of natural sounds in the auditory pathway, A1 must integrate multiple channels of distant frequencies [2]; for this purpose, the disordered tonotopy can be beneficial because a neuron can easily collect information regarding distant (and often harmonically related) frequencies from other cells within its neighbourhood. Our result suggests the existence of a common adaptive strategy underlying V1 and A1, which would be consistent with experimental studies that exchanged the peripheral inputs of the visual and auditory systems and suggested the sensory experiences had a dominant effect on cortical organization [28, 29, 30].

Our final result suggested that a common mechanism may underlie the complex cells of V1 and the pitch-selective cells of A1. Additional support for this notion was provided by recent evidence indicating that the pitch-selective cells are most commonly found in the supragranular layer [27], and V1 complex cells display a similar tendency. It has been hypothesized that V1 complex cells collect information from neighbouring cells that are selective to different phases of similar orientations; in an analogous way, A1 pitch-selective cells could collect information from the activities of neighbouring cells, which in this case could be selective to different frequencies sharing a single $f_0$. To the best of our knowledge, no previous studies in the literature have attempted to use this analogy of V1 complex cells to explain A1 pitch-selective cells (however, other potential analogues have been mentioned [31, 32]). Our results and further investigations should help us to understand these pitch-selective cells from an integrated, computational viewpoint. Another issue that must be addressed is what functional roles the other units in the second layer play. One possible answer to this question may be multipeaked responses related to harmony [3], which have been explained in part by sparse coding [18, 19]; however, this answer has not yet been confirmed by existing evidence and must therefore be assessed in detail by further investigations.

**Acknowledgement**

Supported by KAKENHI (11J04424 for HT; 22650041, 20240020, 23119708 for MO).

# References

[1] D. H. Hubel and T. N. Wiesel. Receptive fields, binocular interaction and functional architecture in the cat's visual cortex. *The Journal of Physiology*, 160(1):106–154, 1962.

[2] C. E. Schreiner, H. L. Read, and M. L. Sutter. Modular organization of frequency integration in primary auditory cortex. *Annual Review of Neuroscience*, 23(1):501–529, 2000.

[3] S. C. Kadia and X. Wang. Spectral integration in A1 of awake primates: Neurons with single- and multipeaked tuning characteristics. *Journal of Neurophysiology*, 89(3):1603–1622, 2003.

[4] S. Bandyopadhyay, S. A. Shamma, and P. O. Kanold. Dichotomy of functional organization in the mouse auditory cortex. *Nature Neuroscience*, 13(3):361–368, 2010.

[5] G. Rothschild, I. Nelken, and A. Mizrahi. Functional organization and population dynamics in the mouse primary auditory cortex. *Nature Neuroscience*, 13(3):353–360, 2010.

[6] S. L. Smith and M. Häusser. Parallel processing of visual space by neighboring neurons in mouse visual cortex. *Nature Neuroscience*, 13(9):1144–1149, 2010.

[7] V. Bonin, M. H. Histed, S. Yurgenson, and R. C. Reid. Local diversity and fine-scale organization of receptive fields in mouse visual cortex. *The Journal of Neuroscience*, 31(50):18506–18521, 2011.

[8] E. F. Evans, H. F. Ross, and I. C. Whitfield. The spatial distribution of unit characteristic frequency in the primary auditory cortex of the cat. *The Journal of Physiology*, 179(2):238–247, 1965.

[9] M. H. Goldstein Jr, M. Abeles, R. L. Daly, and J. McIntosh. Functional architecture in cat primary auditory cortex: tonotopic organization. *Journal of Neurophysiology*, 33(1):188–197, 1970.

[10] A. Hyvärinen, J. Hurri, and P. O. Hoyer. *Natural Image Statistics: A probabilistic approach to early computational vision*. Springer-Verlag London Ltd., 2009.

[11] B. A. Olshausen and D. J. Field. Emergence of simple-cell receptive field properties by learning a sparse code for natural images. *Nature*, 381(6583):607–609, 1996.

[12] A. J. Bell and T. J. Sejnowski. The "independent components" of natural scenes are edge filters. *Vision Research*, 37(23):3327–3338, 1997.

[13] A. Hyvärinen and P. O. Hoyer. A two-layer sparse coding model learns simple and complex cell receptive fields and topography from natural images. *Vision Research*, 41(18):2413–2423, 2001.

[14] A. Hyvärinen, P. O. Hoyer, and M. Inki. Topographic independent component analysis. *Neural Computation*, 13(7):1527–1558, 2001.

[15] L. Ma and L. Zhang. A hierarchical generative model for overcomplete topographic representations in natural images. In *IJCNN 2007*.

[16] D. J. Klein, P. Konig, and K. P. Kording. Sparse spectrotemporal coding of sounds. *EURASIP Journal on Applied Signal Processing*, 2003(7):659–667, 2003.

[17] A. M. Saxe, M. Bhand, R. Mudur, B. Suresh, and A. Y. Ng. Unsupervised learning models of primary cortical receptive fields and receptive field plasticity. In *NIPS 2011*.

[18] H. Terashima and H. Hosoya. Sparse codes of harmonic natural sounds and their modulatory interactions. *Network: Computation in Neural Systems*, 20(4):253–267, 2009.

[19] H. Terashima, H. Hosoya, T. Tani, N. Ichinohe, and M. Okada. Sparse coding of harmonic vocalization in monkey auditory cortex. *Neurocomputing*, doi:10.1016/j.neucom.2012.07.009, in press.

[20] J. H. van Hateren and A. van der Schaaf. Independent component filters of natural images compared with simple cells in primary visual cortex. *Proceedings of the Royal Society of London. Series B: Biological Sciences*, 265(1394):359–366, 1998.

[21] International Phonetic Association. *Handbook of the International Phonetic Association: A Guide to the Use of the International Phonetic Alphabet*. Cambridge: Cambridge University Press, 1999.

[22] M. S. Lewicki. Efficient coding of natural sounds. *Nature Neuroscience*, 5(4):356–363, 2002.

[23] E. C. Smith and M. S. Lewicki. Efficient auditory coding. *Nature*, 439(7079):978–982, 2006.

[24] T. Chi and S. Shamma. NSL Matlab Toolbox. http://www.isr.umd.edu/Labs/NSL/Software.htm, 2003.

[25] M. S. Osmanski and X. Wang. Measurement of absolute auditory thresholds in the common marmoset (callithrix jacchus). *Hearing Research*, 277(1–2):127–133, 2011.

[26] B. C. J. Moore. *An introduction to the psychology of hearing*. London: Emerald Group Publishing Ltd., 5th edition, 2003.

[27] D. Bendor and X. Wang. The neuronal representation of pitch in primate auditory cortex. *Nature*, 436(7054):1161–1165, 2005.

[28] M. Sur, P.E. Garraghty, and A. W. Roe. Experimentally induced visual projections into auditory thalamus and cortex. *Science*, 242(4884):1437–1441, 1988.

[29] A. Angelucci, F. Clascá, and M. Sur. Brainstem inputs to the ferret medial geniculate nucleus and the effect of early deafferentation on novel retinal projections to the auditory thalamus. *The Journal of Comparative Neurology*, 400(3):417–439, 1998.

[30] J. Sharma, A. Angelucci, and M. Sur. Induction of visual orientation modules in auditory cortex. *Nature*, 404(6780):841–847, 2000.

[31] B. Shechter and D. A. Depireux. Nonlinearity of coding in primary auditory cortex of the awake ferret. *Neuroscience*, 165(2):612–620, 2010.

[32] C. A. Atencio, T. O. Sharpee, and C. E. Schreiner. Hierarchical computation in the canonical auditory cortical circuit. *Proceedings of the National Academy of Sciences*, 106(51):21894–21899, 2009.

